# Large Margin Taxonomy Embedding with an Application to Document Categorization

**Kilian Weinberger**
Yahoo! Research
kilian@yahoo-inc.com

**Olivier Chapelle**
Yahoo! Research
chap@yahoo-inc.com

## Abstract

Applications of multi-class classification, such as document categorization, often appear in cost-sensitive settings. Recent work has significantly improved the state of the art by moving beyond "flat" classification through incorporation of class hierarchies [4]. We present a novel algorithm that goes beyond hierarchical classification and estimates the latent semantic space that underlies the class hierarchy. In this space, each class is represented by a prototype and classification is done with the simple nearest neighbor rule. The optimization of the semantic space incorporates large margin constraints that ensure that for each instance the correct class prototype is closer than any other. We show that our optimization is convex and can be solved efficiently for large data sets. Experiments on the OHSUMED medical journal data base yield state-of-the-art results on topic categorization.

## 1 Introduction

Multi-class classification is a problem that arises in many applications of machine learning. In many cases the cost of misclassification varies strongly between classes. For example, in the context of object recognition it may be significantly worse to misclassify a *male pedestrian* as a *traffic light* than as a *female pedestrian*. Similarly, in the context of document categorization it seems more severe to misclassify a medical journal on *heart attack* as a publication on *athlete's foot* than on *Coronary artery disease*. Although the scope of the proposed method is by no means limited to text data and topic hierarchies, for improved clarity we will restrict ourselves to terminology from document categorization throughout this paper.

The most common approach to document categorization is to reduce the problem to a "flat" classification problem [13]. However, it is often the case that the topics are not just discrete classes, but are nodes in a complex taxonomy with rich inter-topic relationships. For example, web pages can be categorized into the Yahoo! web taxonomy or medical journals can be categorized into the Medical Subject Headings (MeSH) taxonomy. Moving beyond flat classification to settings that utilize these hierarchical representations of topics has significantly pushed the state-of-the art [4, 15]. Additional information about inter-topic relationships can for example be leveraged through cost-sensitive decision boundaries or knowledge sharing between documents from closely related classes.

In reality, however, the topic taxonomy is a crude approximation of topic relations, created by an editor with knowledge of the true underlying semantic space of topics. In this paper we propose a method that moves beyond hierarchical presentations and aims to re-discover the continuous latent semantic space underlying the topic taxonomy. Instead of regarding document categorization as classification, we will think of it as a regression problem where new documents are mapped into this latent semantic topic space. Very different from approaches like LSI or LDA [1, 7], our algorithm is entirely supervised and explicitly embeds the topic taxonomy and the documents into a single latent semantic space with "semantically meaningful" Euclidean distances.

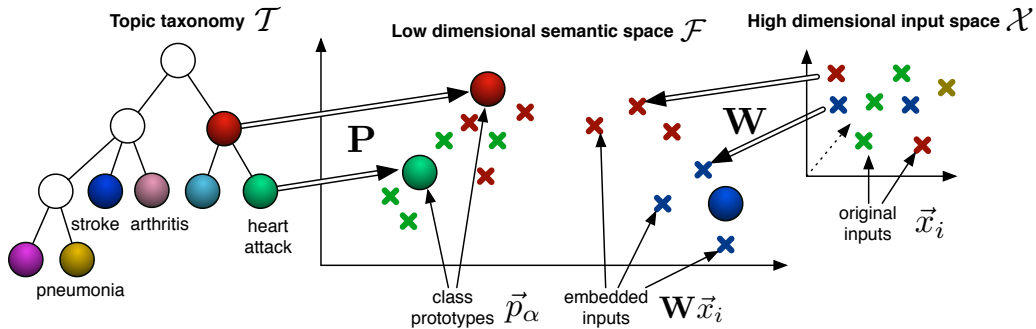

Figure 1: A schematic layout of our taxem method (for Taxonomy Embedding). The classes are embedded as prototypes inside the semantic space. The input documents are mapped into the same space, placed closest to their topic prototypes.

In this paper we derive a method to embed the taxonomy of topics into a latent semantic space in form of topic prototypes. A new document can be classified by first mapping it into this space and then assigning the label of the closest prototype. A key contribution of our paper is the derivation of a convex problem that learns the regressor for the documents and the placement of the prototypes in a single optimization. In particular, it places the topic prototypes such that for each document the prototype of the correct topic is much closer than any other prototype by a large margin. We show that this optimization is a special instance of semi-definite programs [2], that can be solved efficiently [16] for large data sets.

Our paper is structured as follows: In section 2 we introduce necessary notation and a first version of the algorithm based on a two-step approach of first embedding the hierarchical taxonomy into a semantic space and then regressing the input documents close to their respective topic prototypes. In section 3 we extend our model to a single optimization that learns both steps in one convex optimization with large margin constraints. We evaluate our method in section 4 and demonstrate state-of-the-art results on eight different document categorization tasks from the OHSUMED medical journal data set. Finally, we relate our method to previous work in section 5 and conclude in section 6.

## 2   Method

We assume that our input consists of documents, represented as a set of high dimensional sparse vectors $\vec{x}_1, \ldots, \vec{x}_n \in \mathcal{X}$ of dimensionality $d$. Typically, these could be binary bag of words indicators or tfidf scores. In addition, the documents are accompanied by single topic labels $y_1, \ldots, y_n \in \{1, \ldots, c\}$ that lie in some taxonomy $\mathcal{T}$ with $c$ total topics. This taxonomy $\mathcal{T}$ gives rise to some cost matrix $\mathbf{C} \in \mathcal{R}^{c \times c}$, where $\mathbf{C}_{\alpha\beta} \geq 0$ defines the cost of misclassifying an element of topic $\alpha$ as $\beta$ and $\mathbf{C}_{\alpha\alpha} = 0$. Technically, we only require knowledge of the cost matrix $\mathbf{C}$, which could also be obtained from side-information independent of a topic taxonomy. In this paper we will not focus on how $\mathbf{C}$ is obtained. However, we would like to point out that a common way to infer a cost matrix from a taxonomy is to set $\mathbf{C}_{\alpha\beta}$ to the length of the shortest path between node $\alpha$ and $\beta$, but other approaches have also been studied [3].

Throughout this paper we denote document indices as $i, j \in \{1, \ldots, n\}$ and topic indices as $\alpha, \beta \in \{1, \ldots, c\}$. Matrices are written in bold (e.g. $\mathbf{C}$) and vectors have top arrows (e.g. $\vec{x}_i$).

Figure 1 illustrates our setup schematically. We would like to create a low dimensional semantic feature space $\mathcal{F}$ in which we represent each topic $\alpha$ as a topic prototype $\vec{p}_\alpha \in \mathcal{F}$ and each document $\vec{x}_i \in \mathcal{X}$ as a low dimensional vector $\vec{z}_i \in \mathcal{F}$. Our goal is to discover a representation of the data where distances reflect true underlying dissimilarities and proximity to prototypes indicates topic membership. In other words, documents on the same or related topics should be close to the respective topic prototypes, documents on highly different topics should be well separated.

Throughout this paper we will assume that $\mathcal{F} = \mathcal{R}^c$, however our method can easily be adapted to even lower dimensional settings $\mathcal{F} = \mathcal{R}^r$ where $r < c$. As an essential part of our method is to embed the classes that are typically found in a taxonomy, we refer to our algorithm as *taxem* (short for "taxonomy embedding").

**Embedding topic prototypes**

The first step of our algorithm is to embed the document taxonomy into a Euclidean vector space. More formally, we derive topic prototypes $\vec{p}_1, \ldots, \vec{p}_c \in \mathcal{F}$ based on the cost matrix $\mathbf{C}$, where $\vec{p}_\alpha$ is the prototype that represents topic $\alpha$. To simplify notation, we define the matrix $\mathbf{P} = [\vec{p}_1, \ldots, \vec{p}_c] \in \mathcal{R}^{c \times c}$ whose columns consist of the topic prototypes.

There are many ways to derive the prototypes from the cost matrix $\mathbf{C}$. By far the simplest method is to ignore the cost matrix $\mathbf{C}$ entirely and let $\mathbf{P}_I = \mathbf{I}$, where $\mathbf{I} \in \mathcal{R}^{c \times c}$ denotes the identity matrix. This results in a $c$ dimensional feature space, where the class-prototypes are all in distance $\sqrt{2}$ from each other at the corner of a c-dimensional simplex. We will refer to $\mathbf{P}_I$ as the *simplex* prototypes.

Better results can be expected when the prototypes of similar topics are closer than those of dissimilar topics. We use the cost matrix $\mathbf{C}$ as an estimate of dissimilarity and aim to place the prototypes such that the distance $\|\vec{p}_\alpha - \vec{p}_\beta\|_2^2$ reflects the cost specified in $\mathbf{C}_{\alpha\beta}^2$. More formally, we set

$$\mathbf{P}_{mds} = \operatorname{argmin}_{\mathbf{P}} \sum_{\alpha,\beta=1}^{c} (\|\vec{p}_\alpha - \vec{p}_\beta\|_2^2 - (\mathbf{C}_{\alpha\beta})^2)^2. \tag{1}$$

If the cost matrix $\mathbf{C}$ defines Euclidean distances (e.g. when the cost is obtained through the shortest path between nodes), we can solve eq. (1) with metric multi-dimensional scaling [5]. Let us denote $\bar{\mathbf{C}} = -\frac{1}{2}\mathbf{H}\mathbf{C}\mathbf{H}$, where the centering matrix $\mathbf{H}$ is defined as $\mathbf{H} = \mathbf{I} - \frac{1}{c}\mathbf{1}\mathbf{1}^\top$, and let its eigenvector decomposition be $\bar{\mathbf{C}} = \mathbf{V}\Lambda\mathbf{V}^\top$. We obtain the solution by setting $\mathbf{P}_{mds} = \sqrt{\Lambda}\mathbf{V}$. We will refer to $\mathbf{P}_{mds}$ as the *mds* prototypes.[1]

Both prototypes embeddings $\mathbf{P}_I$ and $\mathbf{P}_{mds}$ are still independent of the input data $\{\vec{x}_i\}$. Before we can derive a more sophisticated method to place the prototypes with large margin constraints on the document vectors, we will briefly describe the mapping $\mathbf{W} : \mathcal{X} \to \mathcal{F}$ of the input documents into the low dimensional feature space $\mathcal{F}$.

**Document regression**

Assume for now that we have found a suitable embedding $\mathbf{P}$ of the class-prototypes. We need to find an appropriate mapping $\mathbf{W} : \mathcal{X} \to \mathcal{F}$, that maps each input $\vec{x}_i$ with label $y_i$ as close as possible to its topic prototype $\vec{p}_{y_i}$. We can find such a linear transformation $\vec{z}_i = \mathbf{W}\vec{x}_i$ by setting

$$\mathbf{W} = \operatorname{argmin}_{\mathbf{W}} \sum_i \|\vec{p}_{y_i} - \mathbf{W}\vec{x}_i\|^2 + \lambda\|\mathbf{W}\|_F^2. \tag{2}$$

Here, $\lambda$ is the weight of the regularization of $\mathbf{W}$, which is necessary to prevent potential overfitting due to the high number of parameters in $\mathbf{W}$. The minimization in eq. (2) is an instance of linear ridge regression and has the closed form solution

$$\mathbf{W} = \mathbf{P}\mathbf{J}\mathbf{X}^\top(\mathbf{X}\mathbf{X}^\top + \lambda\mathbf{I})^{-1}, \tag{3}$$

where $\mathbf{X} = [\vec{x}_1, \ldots \vec{x}_n]$ and $\mathbf{J} \in \{0,1\}^{c \times n}$, with $\mathbf{J}_{\alpha i} = 1$ if and only if $y_i = \alpha$. Please note that eq. (3) can be solved very accurately without the need to ever compute the $d \times d$ matrix inverse $(\mathbf{X}\mathbf{X}^\top + \lambda\mathbf{I})^{-1}$ explicitly, by solving with linear conjugate gradient for each row of $\mathbf{W}$ independently.

**Inference**

Given an input vector $\vec{x}_t$ we first map it into $\mathcal{F}$ and estimate its label as the topic with the closest prototype $\vec{p}_\alpha$

$$\hat{y}_t = \operatorname{argmin}_\alpha \|\vec{p}_\alpha - \mathbf{W}\vec{x}_t\|^2. \tag{4}$$

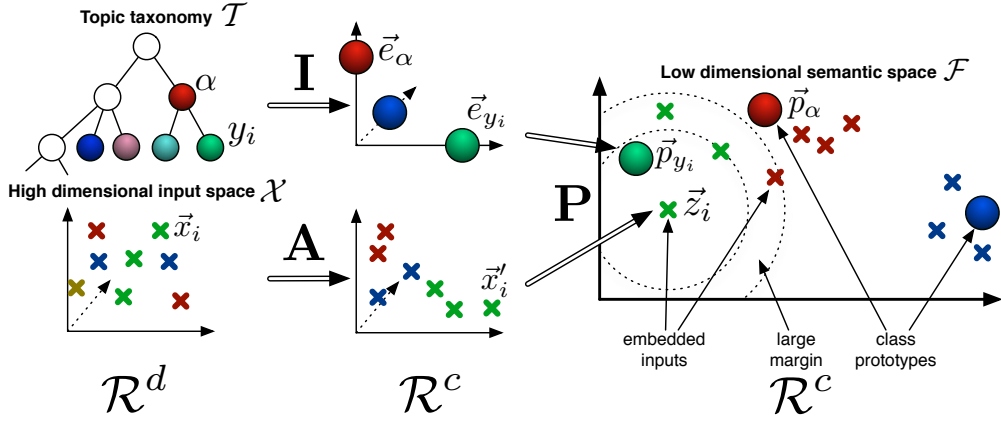

Figure 2: The schematic layout of the large-margin embedding of the taxonomy and the documents. As a first step, we represent topic $\alpha$ as the vector $\vec{e}_\alpha$ and document $\vec{x}_i$ as $\vec{x}_i' = \mathbf{A}\vec{x}_i$. We then learn the matrix $\mathbf{P}$ whose columns are the prototypes $\vec{p}_\alpha = \mathbf{P}\vec{e}_\alpha$ and which defines the final transformation of the documents $\vec{z}_i = \mathbf{P}\vec{x}_i'$. This final transformation is learned such that the correct prototype $\vec{p}_{y_i}$ is closer to $\vec{z}_i$ than any other prototype $\vec{p}_\alpha$ by a large margin.

For a given set of labeled documents $(\vec{x}_1, y_1), \ldots, (\vec{x}_n, y_n)$ we measure the quality of our semantic space with the averaged cost-sensitive misclassification loss,

$$\mathcal{E} = \frac{1}{n} \sum_{i=1}^{n} \mathbf{C}_{y_i \hat{y}_i}. \tag{5}$$

## 3   Large Margin Prototypes

So far we have introduced a two step approach: First, we find the prototypes $\mathbf{P}$ based on the cost matrix $\mathbf{C}$, then we learn the mapping $\vec{x} \to \mathbf{W}\vec{x}$ that maps each input closest to the prototype of its class. However, learning the prototypes independent of the data $\{\vec{x}_i\}$ is far from optimal in order to reduce the loss in (5). In this section we will create a joint optimization problem that places the prototypes $\mathbf{P}$ and learns the mapping $\mathbf{W}$ while minimizing an upper bound on (5).

**Combined learning**
In our attempt to learn both mappings jointly, we are faced with a "chicken and egg" problem. We want to map the input documents closest to their prototypes and at the same time place the prototypes where the documents of the respective topic are mapped to. Therefore our first task is to de-tangle this mutual dependency of $\mathbf{W}$ and $\mathbf{P}$. Let us define $\mathbf{A}$ as the following matrix product:

$$\mathbf{A} = \mathbf{J}\mathbf{X}^\top (\mathbf{X}\mathbf{X}^\top + \lambda \mathbf{I})^{-1}. \tag{6}$$

It follows immediately form eqs. (3) and (6) that $\mathbf{W} = \mathbf{P}\mathbf{A}$. Note that eq. (6) is entirely independent of $\mathbf{P}$ and can be pre-computed before the prototypes have been positioned. With this relation we have reduced the problem of determining $\mathbf{W}$ and $\mathbf{P}$ to the single problem of determining $\mathbf{P}$.

Let $\vec{x}_i' = \mathbf{A}\vec{x}_i$ and let $\vec{e}_\alpha = [0, \ldots, 1, \ldots, 0]^\top$ be the vector with all zeros and a single 1 in the $\alpha^{th}$ position. We can then rewrite both, the topic prototypes $\vec{p}_\alpha$ and the low dimensional documents $\vec{z}_i$, as vectors within the range of $\mathbf{P} : \mathcal{R}^c \to \mathcal{R}^c$:

$$\vec{p}_\alpha = \mathbf{P}\vec{e}_\alpha, \text{ and } \vec{z}_i = \mathbf{P}\vec{x}_i'. \tag{7}$$

**Optimization**
Ideally we would like to learn $\mathbf{P}$ to minimize (5) directly. However, this function is non-continuous and non-differentiable. For this reason we will derive a surrogate loss function that strictly bounds (5) from above.

The loss for a specific document $\vec{x}_i$ is zero if its corresponding vector $\vec{z}_i$ is closer to the correct prototype $\vec{p}_{y_i}$ than to any other prototype $\vec{p}_\alpha$. For better generalization it would be preferable if prototype $\vec{p}_{y_i}$ was in fact much closer by a large margin. We can go even further and demand that prototypes that would incur a larger misclassification loss should be further separated than those with a small cost. More explicitly, we will try to enforce a margin of $\mathbf{C}_{y_i\alpha}$. We can express this condition as a set of "soft" inequality constraints, in terms of squared-distances,

$$\forall i, \alpha \neq y_i \quad \|\mathbf{P}(\vec{e}_{y_i} - \vec{x}'_i)\|_2^2 + \mathbf{C}_{y_i\alpha} \leq \|\mathbf{P}(\vec{e}_\alpha - \vec{x}'_i)\|_2^2 + \xi_{i\alpha}, \tag{8}$$

where the slack-variable $\xi_{i\alpha} \geq 0$ absorbs the amount of violation of prototype $\vec{p}_\alpha$ into the margin of $\vec{x}'_i$. Given this formulation, we create an upper bound on the loss function (5):

**Lemma 1** *Given a prototype matrix* $\mathbf{P}$*, the training error* (5) *is bounded above by* $\frac{1}{n}\sum_{i\alpha}\xi_{i\alpha}$.

**Proof:** First, note that we can rewrite the assignment of the closest prototype (4) as $\hat{y}_i = \mathrm{argmin}_\alpha \|\mathbf{P}(\vec{e}_\alpha - \vec{x}'_i)\|^2$. It follows that $\|\mathbf{P}(\vec{e}_{y_i} - \vec{x}'_i)\|_2^2 - \|\mathbf{P}(\vec{e}_{\hat{y}_i} - \vec{x}'_i)\|_2^2 \geq 0$ for all $i$ (with equality when $\hat{y}_i = y_i$). We therefore obtain:

$$\xi_{i\hat{y}_i} = \|\mathbf{P}(\vec{e}_{y_i} - \vec{x}'_i)\|_2^2 + \mathbf{C}_{y_i\hat{y}_i} - \|\mathbf{P}(\vec{e}_{\hat{y}_i} - \vec{x}'_i)\|_2^2 \geq \mathbf{C}_{y_i\hat{y}_i}. \tag{9}$$

The result follows immediately from (9) and that $\xi_{i\alpha} \geq 0$:

$$\sum_{i,\alpha} \xi_{i\alpha} \geq \sum_i \xi_{i\hat{y}_i} \geq \sum_{i,\hat{y}_i} \mathbf{C}_{y_i\hat{y}_i}. \tag{10}$$

Lemma 1, together with the constraints in eq. (8), allows us to create an optimization problem that minimizes an upper bound on the average loss in eq. (5) with maximum-margin constraints:

$$\begin{array}{l} \textbf{Minimize } \sum_{i,\alpha} \xi_{i\alpha} \textbf{ subject to:} \\ \mathbf{P} \\ \textbf{(1)} \ \|\mathbf{P}(\vec{e}_{y_i} - \vec{x}'_i)\|_2^2 + \mathbf{C}_{y_i\alpha} \leq \|\mathbf{P}(\vec{e}_\alpha - \vec{x}'_i)\|_2^2 + \xi_{i\alpha} \\ \textbf{(2)} \ \xi_{i\alpha} \geq 0 \end{array} \tag{11}$$

Note that if we have a very large number of classes, it might be beneficial to choose $\mathbf{P} \in \mathcal{R}^{r \times c}$ with $r < c$. However, the convex formulation described in the next paragraph requires $\mathbf{P}$ to be square.

**Convex formulation**
The optimization in eq. (11) is not convex. The constraints of type (8) are quadratic with respect to $\mathbf{P}$. Intuitively, any solution $\mathbf{P}$ gives rise to infinitely many solutions as any rotation of $\mathbf{P}$ results in the same objective value and also satisfies all constraints. We can make (11) invariant to rotation by defining $\mathbf{Q} = \mathbf{P}^\top\mathbf{P}$, and rewriting all distances in terms of $\mathbf{Q}$,

$$\|\mathbf{P}(\vec{e}_\alpha - \vec{x}'_i)\|_2^2 = (\vec{e}_\alpha - \vec{x}'_i)^\top \mathbf{Q}(\vec{e}_\alpha - \vec{x}'_i) = \|\vec{e}_\alpha - \vec{x}'_i\|_\mathbf{Q}^2. \tag{12}$$

Note that the distance formulation in eq. (12) is linear with respect to $\mathbf{Q}$. As long as the matrix $\mathbf{Q}$ is positive semi-definite, we can re-decompose it into $\mathbf{Q} = \mathbf{P}^\top\mathbf{P}$. Hence, we enforce positive semi-definiteness of $\mathbf{Q}$ by adding the constraint $\mathbf{Q} \succeq 0$. We can now solve (11) in terms of $\mathbf{Q}$ instead of $\mathbf{P}$ with the large-margin constraints

$$\forall i, \alpha \neq y_i \quad \|\vec{e}_{y_i} - \vec{x}'_i\|_\mathbf{Q}^2 + \mathbf{C}_{y_i\alpha} \leq \|\vec{e}_\alpha - \vec{x}'_i\|_\mathbf{Q}^2 + \xi_{i\alpha}. \tag{13}$$

**Regularization**
If the size of the training data $n$ is small compared to the number of parameters $c^2$, we might run into problems of overfitting to the training data set. To counter those effects, we add a regularization term to the objective function.

Even if the training data might differ from the test data, we know that the taxonomy does not change. It is straight-forward to verify that if the mapping $\mathbf{A}$ was perfect, i.e. for all $i$ we have $\vec{x}'_i = \vec{e}_{y_i}$, $\mathbf{P}_{mds}$ satisfies all constraints (8) as equalities with zero slack. This gives us confidence that the optimal solution $\mathbf{P}$ for the *test* data should not deviate too much from $\mathbf{P}_{mds}$. We will therefore penalize

| Top category | A | B | C | D | E | F | G | H |
|---|---|---|---|---|---|---|---|---|
| # samples n | 7544 | 4772 | 4858 | 2701 | 7300 | 1961 | 8694 | 8155 |
| # topics c | 424 | 160 | 453 | 339 | 457 | 151 | 425 | 150 |
| # nodes | 519 | 312 | 610 | 608 | 559 | 218 | 533 | 170 |

Table 1: Statistics of the different OHSUMED problems. Note that not all nodes are populated and that we pruned all strictly un-populated subtrees.

$\|\mathbf{Q} - \bar{\mathbf{C}}\|_F^2$, where $\bar{\mathbf{C}} = \mathbf{P}_{mds}^\top \mathbf{P}_{mds}$ (as defined in section 2). The final convex optimization of taxem with regularized objective becomes:

$$
\begin{aligned}
&\textbf{Minimize } (1-\mu) \sum_{i,\alpha} \xi_{i\alpha} + \mu \|\mathbf{Q} - \bar{\mathbf{C}}\|_F^2 \textbf{ subject to:} \\
&\mathbf{Q} \\
&\textbf{(1) } \|\vec{e}_{y_i} - \vec{x}_i'\|_\mathbf{Q}^2 + \mathbf{C}_{y_i\alpha} \leq \|\vec{e}_\alpha - \vec{x}_i'\|_\mathbf{Q}^2 + \xi_{i\alpha} \\
&\textbf{(2) } \xi_{i\alpha} \geq 0 \\
&\textbf{(3) } \mathbf{Q} \succeq 0
\end{aligned}
\tag{14}
$$

The constant $\mu \in [0,1]$ regulates the impact of the regularization term. The optimization in (14) is an instance of a semidefinite program (SDP) [2]. Although SDPs can often be expensive to solve, the optimization (14) falls into a special category[2] and can be solved very efficiently with special purpose sub-gradient solvers even with millions of constraints [16]. Once the optimal solution $\mathbf{Q}^*$ is found, one can obtain the position of the prototypes with a simple svd or cholesky decomposition $\mathbf{Q}^* = \mathbf{P}^\top \mathbf{P}$ and consequently also obtains the mapping $\mathbf{W}$ from $\mathbf{W} = \mathbf{PA}$.

## 4 Results

We evaluated our algorithm taxem on several classification problems derived from categorizing publications in the public OHSUMED medical journal data base into the Medical Subject Headings (MeSH) taxonomy.

**Setup and data set description**
We used the OHSUMED 87 corpus [9], which consists of abstracts and titles of medical publications. Each of these entries has been assigned one or more categories in the MeSH taxonomy[3]. We used the 2001 version of these MeSH headings resulting in about 20k categories organized in a taxonomy. To preprocess the data we proceeded as follows: First, we discarded all database entries with empty abstracts, which left us with 36890 documents. We tokenized (after stop word removal and stemming) each abstract, and represented the corresponding bag of words as its $d = 60727$ dimensional tfidf scores (normalized to unit length). We removed all topic categories that did not appear in the MeSH taxonomy (due to out-dated topic names). We further removed all subtrees of nodes that were populated with one or less documents. The top categories in the OHSUMED data base are "orthogonal" — for instance the B top level category is about organism while C is about diseases. We thus created 8 independent classification problems out of the top categories A,B,C,D,E,F,G,H. For each problem, we kept only the abstracts that were assigned exactly one category in that tree, making each problem single-label. The statistics of the different problems are summrized in Table 1. For each problem, we created a $70\%/30\%$ random split in training and test samples, ensuring however that each topic had at least one document in the training corpus.

**Document Categorization**
The classification results on the OHSUMED data set are summarized in Table 2. We set the regularization constants to be $\lambda = 1$ for the regression and $\mu = 0.1$ for the SDP. Preliminary experiments on data set B showed that regularization was important but the exact settings of the $\mu$ and $\lambda$ had no

| data | SVM 1/all | MCSVM | SVM cost | SVM tax | $\mathbf{P}_I$-taxem | $\mathbf{P}_{mds}$-taxem | LM-taxem |
|------|-----------|-------|----------|---------|----------------------|--------------------------|----------|
| A | 2.17 | 2.13 | 2.11 | **1.96** | 2.11 | 2.33 | **1.95** |
| B | 1.50 | **1.38** | 1.64 | 1.52 | 1.57 | 1.99 | **1.39** |
| C | 2.41 | 2.32 | **2.25** | **2.25** | 2.30 | 2.61 | **2.16** |
| D | 3.10 | **2.76** | 2.92 | **2.82** | **2.82** | 3.05 | **2.66** |
| E | 3.44 | 3.42 | 3.26 | 3.25 | 3.45 | **3.05** | **3.05** |
| F | **2.59** | **2.65** | **2.66** | 2.69 | **2.63** | 2.77 | **2.51** |
| G | 3.98 | 4.12 | 3.89 | 3.82 | 4.10 | **3.63** | **3.59** |
| H | 2.42 | 2.48 | 2.40 | 2.32 | 2.45 | **2.24** | **2.17** |
| all | 2.78 | 2.77 | 2.77 | 2.65 | 2.79 | 2.73 | **2.50** |

Table 2: The cost-sensitive test error results on various ohsumed classification data sets. The algorithms are from left to right: one vs. all SVM, MCSVM [6], cost-sensitive MCSVM, Hierarchical SVM [4], simplex regression, mds regression, large-margin taxem. The best results (up to statistical significance) are highlighted in bold. The taxem algorithm obtains the lowest overall loss and the lowest individual loss on each data set except B.

crucial impact. We compared taxem against four commonly used algorithms for document categorization: 1. A linear support vector machine (SVM) trained in one vs. all mode (SVM 1/all) [12], 2. the Crammer and Singer multi-class SVM formulation (MCSVM) [6], 3. the Cai and Hoffmann SVM classifer with cost-sensitive loss function (SVM cost) [4], 4. the Cai and Hoffmann SVM formulation with a cost sensitive hierarchical loss function (SVM tax) [4]. All SVM classifiers were trained with regularization constant $C = 1$ (which worked best on problem B; this value is also commonly used in text classification when the documents have unit length). Further, we also evaluated the difference between our large margin formulation (taxem) and the results with the simplex ($\mathbf{P}_I$-taxem) and mds ($\mathbf{P}_{mds}$-taxem) prototypes. To check the significance of our results we applied a standard t-test with a $5\%$ confidence interval. The best results up to statistical significance are highlighted in bold font. The final entry in Table 2 shows the average error over all test points in all data sets. Up to statistical significance, taxem obtains the lowest loss on all data sets and the lowest overall loss. Ignoring statistical significance, taxem has the lowest loss on all data sets except B. All algorithms had comparable speed during test-time. The computation time required for solving eq. (6) and the optimization (14) was on the order of several minutes with our MATLAB$^{\text{TM}}$ implementation on a standard Intel$^{\text{TM}}$ 1.8GHz core 2 duo processor (without parallelization efforts).

# 5   Related Work

In recent years, several algorithms for document categorization have been proposed. Several authors proposed adaptations of support vector machines that incorporate the topic taxonomy through cost-sensitive loss re-weighting and classification at multiple nodes in the hierarhchy [4, 8, 11]. Our algorithm is based on a very different intuition. It differs from all these methods in that it learns a low dimensional semantic representation of the documents and classifies by finding the nearest prototype.

Most related to our work is probably the work by Karypis and Han [10]. Although their algorithm also reduces the dimensionality with a linear projection, their low dimensional space is obtained through supervised clustering on the document data. In contrast, the semantic space obtained with taxem is obtained through a convex optimization with maximum margin constraints. Further, the low dimensional representation of our method is explicitly constructed to give rise to meaningful Euclidean distances.

The optimization with large-margin constraints was partially inspired by recent work on large margin distance metric learning for nearest neighbor classification [16]. However our formulation is a much more light-weight optimization problem with $O(cn)$ constraints instead of $O(n^2)$ as in [16]. The optimization problem in section 3 is also related to recent work on automated speech recognition through discriminative training of Gaussian mixture models [14].

# 6 Conclusion

In this paper, we have presented a novel framework for classification with inter-class relationships based on taxonomy embedding and supervised dimensionality reduction. We derived a single convex optimization problem that learns an embedding of the topic taxonomy as well as a linear mapping from the document space to the resulting low dimensional semantic space.

As future work we are planning to extend our algorithm to the more general setting of document categorization with multiple topic memberships and multi-modal topic distributions. Further, we are keen to explore the implications of our proposed conversion of discrete topic taxonomies into continuous semantic spaces. This framework opens new interesting directions of research that go beyond mere classification. A natural step is to consider the document matching problem (e.g. of web pages and advertisements) in the semantic space: a fast nearest neighbor search can be performed in a joint low dimensional space without having to resort to classification all together.

Although this paper is presented in the context of document categorization, it is important to emphasize that our method is by no means limited to text data or class hierarchies. In fact, the proposed algorithm can be applied in almost all multi-class settings with cost-sensitive loss functions (e.g. object recognition in computer vision).

## Footnotes

[1]If $\bar{\mathbf{C}}$ does not contain Euclidean distances one can use the common approximation of forcing negative eigenvalues in $\Lambda$ to zero and thereby fall back onto the projection of $\mathbf{C}$ onto the cone of positive semi-definite matrices.

[2]The solver described in [16] utilizes that many constraints are inactive and that the sub-gradient can be efficiently derived from previous gradient steps.

[3]see http://en.wikipedia.org/wiki/Medical_Subject_Headings

## References

[1] D. Blei, A. Ng, M. Jordan, and J. Lafferty. Latent Dirichlet Allocation. *Journal of Machine Learning Research*, 3(4-5):993–1022, 2003.

[2] S. Boyd and L. Vandenberghe. *Convex Optimization*. Cambridge University Press, 2004.

[3] A. Budanitsky and G. Hirst. Semantic distance in wordnet: An experimental, application-oriented evaluation of five measures. In *Workshop on WordNet and Other Lexical Resources, in the North American Chapter of the Association for Co mputational Linguistics (NAACL)*, 2001.

[4] L. Cai and T. Hofmann. Hierarchical document categorization with support vector machines. In *ACM 13th Conference on Information and Knowledge Management*, 2004.

[5] T. Cox and M. Cox. *Multidimensional Scaling*. Chapman & Hall, London, 1994.

[6] K. Crammer and Y. Singer. On the algorithmic implementation of multiclass kernel-based vector machines. *Journal of Machine Learning Research*, 2:265–292, 2001.

[7] S. Deerwester, S. Dumais, G. Furnas, T. Landauer, and R. Harshman. Indexing by latent semantic analysis. *Journal of the American Society for Information Science*, 41(6):391–407, 1990.

[8] S. Dumais and H. Chen. Hierarchical classification of Web content. In *Proceedings of SIGIR-00, 23rd ACM International Conference on Research and Development in Information Retrieval*, pages 256–263. ACM Press, New York, US, 2000.

[9] W. Hersh, C. Buckley, T. J. Leone, and D. Hickam. OHSUMED: an interactive retrieval evaluation and new large test collection for research. In *SIGIR '94: Proceedings of the 17th annual international ACM conference on Research and development in information retrieval*, pages 192–201. Springer-Verlag New York, Inc., 1994.

[10] G. Karypis, E. Hong, and S. Han. Concept indexing a fast dimensionality reduction algorithm with applications to document retrieval & categorization, 2000. Technical Report: 00-016 karypis, han@cs.umn.edu Last updated on.

[11] T.-Y. Liu, Y. Yang, H. Wan, H.-J. Zeng, Z. Chen, and W.-Y. Ma. Support vector machines classification with a very large-scale taxonomy. *SIGKDD Explorations Newsletter*, 7(1):36–43, 2005.

[12] R. Rifkin and A. Klautau. In Defense of One-Vs-All Classification. *The Journal of Machine Learning Research*, 5:101–141, 2004.

[13] F. Sebastiani. Machine learning in automated text categorization. *ACM Computing Surveys*, 34(1):1–47, 2002.

[14] F. Sha and L. K. Saul. Large margin hidden markov models for automatic speech recognition. In *Advances in Neural Information Processing Systems 19*, Cambridge, MA, 2007. MIT Press.

[15] A. Weigend, E. Wiener, and J. Pedersen. Exploiting Hierarchy in Text Categorization. *Information Retrieval*, 1(3):193–216, 1999.

[16] K. Q. Weinberger and L. K. Saul. Fast solvers and efficient implementations for distance metric learning. In *Proceedings of the Twenty-fifth International Conference on Machine Learning (ICML 2008)*, Helsinki, Finland, 2008.

